# Foraging in an Uncertain Environment Using Predictive Hebbian Learning

**P. Read Montague** * **Peter Dayan, and Terrence J. Sejnowski**
Computational Neurobiology Lab, The Salk Institute,
10010 N. Torrey Pines Rd,
La Jolla, CA, 92037, USA
read@bohr.bcm.tmc.edu

## Abstract

Survival is enhanced by an ability to predict the availability of food, the likelihood of predators, and the presence of mates. We present a concrete model that uses diffuse neurotransmitter systems to implement a predictive version of a Hebb learning rule embedded in a neural architecture based on anatomical and physiological studies on bees. The model captured the strategies seen in the behavior of bees and a number of other animals when foraging in an uncertain environment. The predictive model suggests a unified way in which neuromodulatory influences can be used to bias actions and control synaptic plasticity.

Successful predictions enhance adaptive behavior by allowing organisms to prepare for future actions, rewards, or punishments. Moreover, it is possible to improve upon behavioral choices if the consequences of executing different actions can be reliably predicted. Although classical and instrumental conditioning results from the psychological literature [1] demonstrate that the vertebrate brain is capable of reliable prediction, how these predictions are computed in brains is not yet known.

The brains of vertebrates and invertebrates possess small nuclei which project axons throughout large expanses of target tissue and deliver various neurotransmitters such as dopamine, norepinephrine, and acetylcholine [4]. The activity in these systems may report on reinforcing stimuli in the world or may reflect an expectation of future reward [5, 6, 7, 8].

A particularly striking example is that of the honeybee. Honeybees can be conditioned to a sensory stimulus such as a color, visual pattern, or an odorant when the sensory stimulus is paired with application of sucrose to the antennae or proboscis. An identified neuron, VUMmx1, projects widely throughout the entire bee brain, becomes active in response to sucrose, and its firing can substitute for the unconditioned odor stimulus in classical conditioning experiments [8]. Similar diffusely projecting neurons in the bee brain may substitute for reward when paired with a visual stimulus.

In this paper, we suggest a role for diffuse neurotransmitter systems in learning and behavior that is analogous to the function we previously postulated for them in developmental self-organization[3, 2]. Specifically, we: (*i*) identify a neural substrate/architecture which is known to exist in both vertebrates and invertebrates and which delivers information to widespread regions of the brain; (*ii*) describe an algorithm that is both mathematically sound and biologically feasible; and (*iii*) show that a version of this local algorithm, in the context of the neural architecture, reproduces the foraging and decision behavior observed in bumble bees and a number of other animals.

Our premise is that the predictive relationships between sensory stimuli and rewards are constructed through these diffuse systems and are used to shape both ongoing behavior and reward-dependent synaptic plasticity. We illustrate this using a simple example from the ethological literature for which constraints are available at a number of different levels.

## A Foraging Problem

Real and colleagues [9, 10] performed a series of experiments on bumble bees foraging on artificial flowers whose colors, blue and yellow, predicted of the delivery of nectar. They examined how bees respond to the mean and variability of this reward delivery in a foraging version of a stochastic two-armed bandit problem [11]. All the blue flowers contained $2\mu l$ of nectar, $\frac{1}{3}$ of the yellow flowers contained $6\mu l$, and the remaining $\frac{2}{3}$ of the yellow flowers contained no nectar at all. In practice, 85% of the bees' visits were to the constant yield blue flowers despite the equivalent mean return from the more variable yellow flowers. When the contingencies for reward were reversed, the bees switched their preference for flower color within 1 to 3 visits to flowers. They further demonstrated that the bees could be induced to visit the variable and constant flowers with equal frequency if the mean reward from the variable flower type was made sufficiently high.

This experimental finding shows that bumble bees, like honeybees, can learn to associate color with reward. Further, color and odor learning in honeybees has approximately the same time course as the shift in preference described above for the bumble bees [12]. It also indicates that under the conditions of a foraging task, bees prefer less variable rewards and compute the reward availability in the short term. This is a behavioral strategy utilized by a variety of animals under similar conditions for reward [9, 10, 13] suggesting a common set of constraints in the underlying neural substrate.

## The Model

Fig. 1 shows a diagram of the model architecture, which is based on the considerations above about diffuse systems. Sensory input drives the units 'B' and 'Y' representing blue and yellow flowers. These neurons (outputs $x_t^B$ and $x_t^Y$ respectively at time t) project

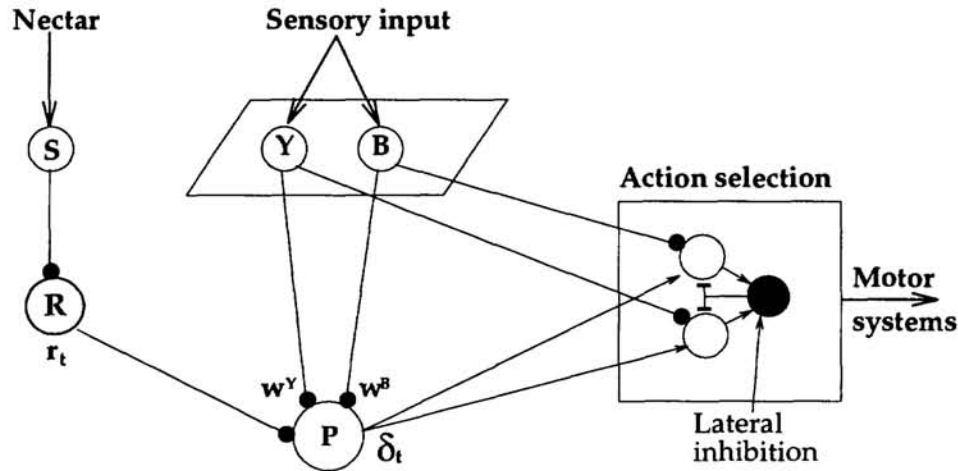

Figure 1: **Neural architecture showing how predictions about future expected rein-forcement can be made in the brain using a diffuse neurotransmitter system** [3, 2]. In the context of bee foraging [9], sensory input drives the units B and Y representing blue and yellow flowers. These units project to a reinforcement neuron P through a set of variable weights (filled circles $w^B$ and $w^Y$) and to an action selection system. Unit S provides input to R and fires while the bee sips the nectar. R projects its output $r_t$ through a fixed weight to P. The variable weights onto P implement predictions about future reward $r_t$ (see text) and P's output is sensitive to temporal changes in its input. The output projections of P, $\delta_t$ (lines with arrows), influence learning and also the selection of actions such as steering in flight and landing, as in equation 5 (see text). Modulated lateral inhibition (dark circle) in the action selection layer symbolizes this. Before encountering a flower and its nectar, the output of P will reflect the temporal difference only between the sensory inputs B and Y. During an encounter with a flower and nectar, the prediction error $\delta_t$ is determined by the output of B or Y and R, and learning occurs at connections $w^B$ and $w^Y$. These strengths are modified according to the correlation between presynaptic activity and the prediction error $\delta_t$ produced by neuron P as in equation 3 (see text). Learning is restricted to visits to flowers [14].

through excitatory connection weights both to a diffusely projecting neuron P (weights $w^B$ and $w^Y$) and to other processing stages which control the selection of actions such as steering in flight and landing. P receives additional input $r_t$ through unchangeable weights. In the absence of nectar ($r_t = 0$), the net input to P becomes $V_t \equiv w_t \cdot x_t = w_t^B x_t^B + w_t^Y x_t^Y$.

The first assumption in the construction of this model is that learning (adjustment of weights) is contingent upon approaching and landing on a flower. This assumption is supported specifically by data from learning in the honeybee: color learning for flowers is restricted to the final few seconds prior to landing on the flower and experiencing the nectar [14].

This fact suggests a simple model in which the strengths of variable connections $w_t$ are adjusted according to a presynaptic correlational rule:

$$\Delta w_t = \alpha x_t r_t \tag{1}$$

where $\alpha$ is the learning rate [15]. There are two problems with this formulation: (*i*) learning would only occur about contingencies in the presence of a reinforcing stimulus ($r_t \neq 0$);

**A**        **B**

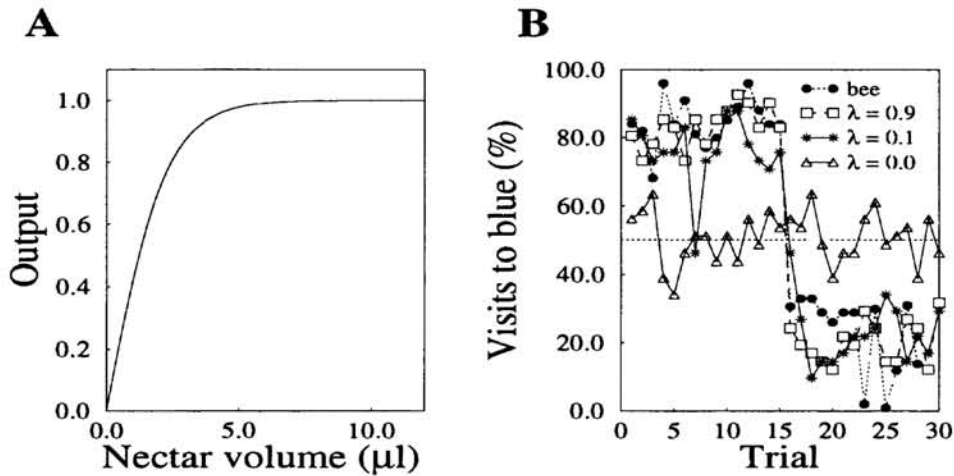

Figure 2: **Simulations of bee foraging behavior using predictive Hebbian learning. A)** Reinforcement neuron output as a function of nectar volume for a fixed concentration of nectar[9, 10]. **B)** Proportion of visits to blue flowers. Each trial represents approximately 40 flower visits averaged over 5 real bees and exactly 40 flower visits for a single model bee. Trials 1 − 15 for the real and model bees had blue flowers as the constant type, the remaining trials had yellow flowers as constant. At the beginning of each trial, $w^Y$ and $w^B$ were set to 0.5 consistent with evidence that information from past foraging bouts is not used[14]. The real bees were more variable than the model bees – sources of stochasticity such as the two-dimensional feeding ground were not represented. The real bees also had a slight preference for blue flowers [21]. Note the slower drop for $\lambda = 0.1$ when the flowers are switched.

and (*ii*) there is no provision for allowing a sensory event to predict the future delivery of reinforcement. The latter problem makes equation 1 inconsistent with a substantial volume of data on classical and instrumental conditioning [16]. Adding a postsynaptic factor to equation 1 does not alter these conclusions [17].

This inadequacy suggests that another form of learning rule and a model in which P has a direct input from $r_t$. Assume that the firing rate of P is sensitive only to changes in its input over time and habituates to constant or slowly varying input, like magnocellular ganglion cells in the retina [18]. Under this assumption, the output of P, $\delta_t$, reflects a temporal derivative of its net input, approximated by:

$$\delta_t = \gamma(r_t + V_t) - (r_{t-1} + V_{t-1}) \tag{2}$$

where $\gamma$ is a factor that controls the weighting of near against distant rewards. We take $\gamma = 1$ for the current discussion.

In the presence of the reinforcement, the weights $w^B$ and $w^Y$ are adjusted according to the simple correlational rule:

$$\Delta w_t = \alpha x_t \delta_t. \tag{3}$$

This permits the weights onto P to act as predictions of the expected reward consequent on landing on a flower and can also be derived in a more general way for the prediction of future values of any scalar quantity [19].

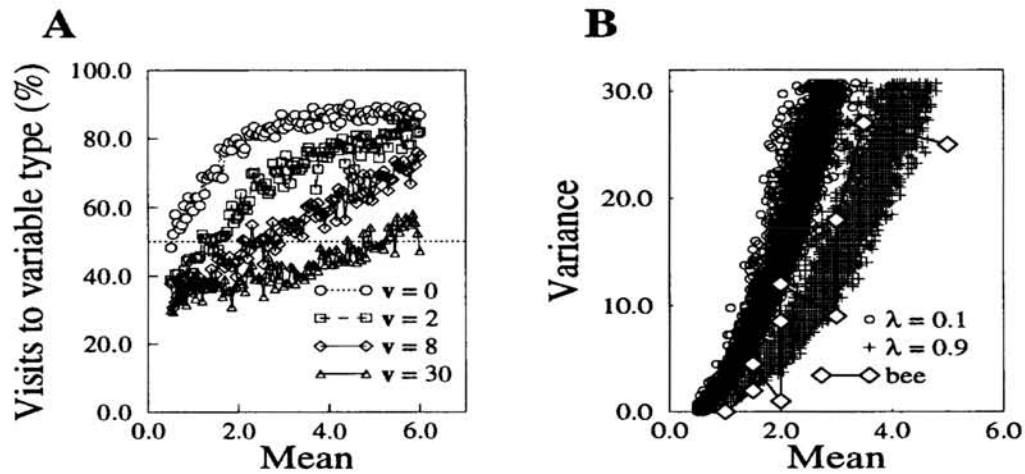

Figure 3: **Tradeoff between the mean and variance of nectar delivery.** A) Method of selecting indifference points. The indifference point is taken as the first mean for a given variance (bold **v** in legend) for which a stochastic trial demonstrates the indifference. This method of calculation tends to bias the indifference points to the left. **B**) Indifference plot for model and real bees. Each point represents the (mean, variance) pair for which the bee sampled each flower type equally. The circles are for $\lambda = 0.1$ and the pluses are for $\lambda = 0.9$.

When the bee actually lands on a flower and samples the nectar, R influences the output of P through its fixed connection (Fig. 1). Suppose that just prior to sampling the nectar the bee switched to viewing a blue flower, for example. Then, since $r_{t-1} = 0$, $\delta_t$ would be $r_t - x^B_{t-1} w^B_{t-1}$. In this way, the term $x^B_{t-1} w^B_{t-1}$ is a prediction of the value of $r_t$ and the difference $r_t - x^B_{t-1} w^B_{t-1}$ is the error in that prediction. Adjusting the weight $w^B_t$ according to the correlational rule in equation 3 allows the weight $w^B_t$, through P's outputs, to report to the rest of the brain the amount of reinforcement $r_t$ expected from blue flowers when they are sensed.

As the model bee flies between flowers, reinforcement from nectar is not present ($r_t = 0$) and $\delta_t$ is proportional to $V_t - V_{t-1}$. $w^B$ and $w^Y$ can again be used as predictions but through modulation of action choice. For example, suppose the learning process in equation 3 sets $w^Y$ less than $w^B$. In flight, switching from viewing yellow flowers to viewing blue flowers causes $\delta_t$ to be positive and biases the activity in any action selection units driven by outgoing connections from B. This makes the bee more likely than chance to land on or steer towards blue flowers. This discussion is not offered as an accurate model of action choice, rather, it simply indicates how output from a diffuse system could also be used to influence action choice.

The biological assumptions of this neural architecture are explicit: *(i)* the diffusely projecting neuron changes its firing according to the temporal difference in its inputs; *(ii)* the output of P is used to adjust its weights upon landing; and *(iii)* the output otherwise biases the selection of actions by modulating the activity of its target neurons.

For the particular case of the bee, both the learning rule described in equation 3 and the biasing of action selection described above can be further simplified for the purposes of a

simple demonstration. As mentioned above, significant learning about a particular flower color may occur only in the $1-2$ seconds just prior to an encounter [21, 14]. This is tantamount to restricting weight changes to each encounter with the reinforcer which allows only the sensory input just preceding the delivery or non-delivery of $r_t$ to drive synaptic plasticity. We therefore make the learning rule punctate, updating the weights on a flower by flower basis. During each encounter with the reinforcer in the environment, P produces a prediction error $\delta_t = r_t - V_{t-1}$ where $r_t$ is the actual reward at time t, and the last flower color seen by the bee at time t, say blue, causes a prediction $V_{t-1} = w^B_{t-1}x^B_{t-1}$ of future reward $r_t$ to be made through the weight $w^B_{t-1}$ and the input activity $x^B_{t-1}$. The weights are then updated using a form of the delta rule[20]:

$$w_t = w_{t-1} + \lambda\delta_t x_{t-1}, \tag{4}$$

where $\lambda$ is a time constant and controls the rate of forgetting. In this rule, the weights from the sensory input onto P still mediate a prediction of r; however, the temporal component for choosing how to steer and when to land has been removed.

We model the temporal biasing of actions such as steering and landing with a probabilistic algorithm that uses the same weights onto P to choose which flower is actually visited on each trial. At each flower visit, the predictions are used directly to choose an action, according to:

$$q(Y) = \frac{e^{\mu(w^Y x^Y)}}{e^{\mu(w^B x^B)} + e^{\mu(w^Y x^Y)}} \tag{5}$$

where $q(Y)$ is the probability of choosing a yellow flower. Values of $\mu > 0$ amplify the difference between the two predictions so that larger values of $\mu$ make it more likely that the larger prediction will result in choice toward the associated flower color. In the limit as $\mu \to \infty$ this approaches a winner-take-all rule. In the simulations, $\mu$ was varied from 2.8 to 6.0 and comparable results obtained. Changing $\mu$ alters the magnitude of the weights that develop onto neuron P since different values of $\mu$ enforce different degrees of competition between the predictions.

To apply the model to the foraging experiment, it is necessary to specify how the amount of nectar in a particular flower gets reported to P. We assume that the reinforcement neuron R delivers its signal $r_t$ as a saturating function of nectar volume (Fig. 2A). Harder and Real [10] suggest just this sort of decelerating function of nectar volume and justify it on biomechanical grounds. Fig. 2B shows the behavior of model bees compared with that of real bees [9] in the experiment testing the extent to which they prefer a constant reward to a variable reward of the same long-term mean. Further details are presented in the figure legend.

The behavior of the model matched the observed data for $\lambda = 0.9$ suggesting that the real bee utilizes information over a small time window for controlling its foraging [9]. At this value of $\lambda$, the average proportion of visits to blue was 85% for the real bees and 83% for the model bees. The constant and variable flower types were switched at trial 15 and both bees switched flower preference in $1-3$ subsequent visits. The average proportion of visits to blue changed to 23% and 20%, respectively, for the real and model bee. Part of the reason for the real bees' apparent preference for blue may come from inherent biases. Honey bees, for instance, are known to learn about shorter wavelengths more quickly than others [21]. In our model, $\lambda$ is a measure of the length of time over which an observation exerts an influence on flower selection rather than being a measure of the bee's time horizon in terms of the mean rate of energy intake [9, 10].

Real bees can be induced to forage equally on the constant and variable flower types if the mean reward from the variable type is made sufficiently large, as in Fig. 3B. For a given variance, the mean reward was increased until the bees appeared indifferent between the flowers. In this experiment, the constant flower type contained $0.5\mu l$ of nectar. The data for the real bee is shown as points connected by a solid line in order to make clear the envelope of the real data. The indifference points for $\lambda = 0.1$ (circles) and $\lambda = 0.9$ (pluses) also demonstrate that a higher value of $\lambda$ is again better at reproducing the bee's behavior. The model captured both the functional relationship and the spread of the real data.

The diffuse neurotransmitter system reports prediction errors to control learning and bias the selection of actions. Distributing such a signal diffusely throughout a large set of target structures permits this prediction error to influence learning generally as a factor in a correlational or Hebbian rule. The same signal, in its second role, biases activity in an action selection system to favor rewarding behavior. In the model, construction of the prediction error only requires convergent input from sensory representations onto a neuron or neurons whose output is a temporal derivative of its input. The output of this neuron can also be used as a secondary reinforcer to associate other sensory stimuli with the predicted reward. We have shown how this relatively simple predictive learning system closely simulates the behavior of bumble bees in a foraging task.

### Acknowledgements

This work was supported by the Howard Hughes Medical Institute, the National Institute of Mental Health, the UK Science and Engineering Research Council, and computational resources from the San Diego Supercomputer Center. We would like to thank Patricia Churchland, Anthony Dayan, Alexandre Pouget, David Raizen, Steven Quartz and Richard Zemel for their helpful comments and criticisms.

## Footnotes

*Division of Neuroscience, Baylor College of Medicine, Houston, TX 77030

## References

[1] Konorksi, J. *Conditioned reflexes and neuron organization,* (Cambridge, England, Cambridge University Press, 1948).

[2] Quartz, SR, Dayan, P, Montague, PR, Sejnowski, TJ. (1992) *Society for Neurosciences Abstracts.* **18**, 210.

[3] Montague, PR, Dayan, P, Nowlan, SJ, Pouget, A, Sejnowski, TJ. (1993) In *Advances in Neural Information Processing Systems 5,* SJ Hanson, JD Cowan, CL Giles, editors, (San Mateo CA: Morgan Kaufmann), pp. 969-976.

[4] Morrison, JH and Magistretti, PJ. *Trends in Neurosciences, 6,* 146 (1983).

[5] Wise, RA. *Behavioral and Brain Sciences, 5,* 39 (1982).

[6] Cole, BJ and Robbins, TW. *Neuropsychopharmacology, 7,* 129 (1992).

[7] Schultz, W. *Seminars in the Neurosciences, 4,* 129 (1992).

[8] Hammer, M, thesis, FU Berlin (1991).

[9] Real, LA. *Science, 253,* pp 980 (1991).

[10] Real, LA. *Ecology,* **62,** 20 (1981); Harder, LD and Real, LA. *Ecology,* **68**(4), 1104 (1987); Real, LA, Ellner, S, Harder, LD. *Ecology,* **71**(4), 1625 (1990).

[11] Berry, DA and Fristedt, B. *Bandit Problems: Sequential Allocation of Experiments.* (London, England: Chapman and Hall, 1985).

[12] Gould, JL. In *Foraging Behavior,* AC Kamil, JR Krebs and HR Pulliam, editors, (New York, NY: Plenum, 1987), p 479.

[13] Krebs, JR, Kacelnik, A, Taylor, P. *Nature,,* **275,** 27 (1978), Houston, A, Kacelnik, A, McNamara, J. In *Functional Ontogeny,* D McFarland, editor, (London: Pitman, 1982).

[14] Menzel, R and Erber, J. *Scientific American,* **239**(1), 102.

[15] Carew, TJ, Hawkins RD, Abrams TW and Kandel ER. *Journal of Neuroscience,* **4**(5), 1217 (1984).

[16] Mackintosh, NJ. *Conditioning and Associative Learning.* (Oxford, England: Oxford University Press, 1983). Sutton, RS and Barto, AG. *Psychological Review,* **88** 2, 135 (1981). Sutton, RS and Barto, AG. *Proceedings of the Ninth Annual Conference of the Cognitive Science Society.* Seattle, WA (1987).

[17] Reeke, GN, Jr and Sporns, O. *Annual Review of Neuroscience.* **16,** 597 (1993).

[18] Dowling, JE. *The Retina.* (Cambridge, MA: Harvard University Press, 1987).

[19] The overall algorithm is a temporal difference (TD) learning rule and is related to an algorithm Samuel devised for teaching a checker playing program, Samuel, AL. *IBM Journal of Research and Development,* **3,** 211 (1959). It was first suggested in its present form in Sutton, RS, thesis, University of Massachusetts (1984); Sutton and Barto [1] showed how it could be used for classical conditioning; Barto, AG, Sutton, RS and Anderson, CW. *IEEE Transactions on Systems, Man, and Cybernetics,* **13,** 834 (1983) used a variant of it in a form of instrumental conditioning task; Barto, AG, Sutton, RS, Watkins, CJCH, *Technical Report 89-95,* (Computer and Information Science, University of Massachusetts, Amherst, MA, 1989); Barto, AG, Bradtke, SJ, Singh, SP, *Technical Report 91-57,* (Computer and Information Science, University of Massachusetts, Amherst, MA, 1991) showed its relationship to dynamic programming, an engineering method of optimal control.

[20] Rescorla, RA and Wagner, AR. In *Classical Conditioning II: Current Research and Theory,* AH Black and WF Prokasy, editors, (New York, NY: Appleton-Century-Crofts, 1972), p 64; Widrow, B and Stearns, SD. *Adaptive Signal Processing,* (Englewood Cliffs, NJ: Prentice-Hall, 1985).

[21] Menzel, R, Erber, J and Masuhr, J. In *Experimental Analysis of Insect Behavior,* LB Browne, editor, (Berlin, Germany: Springer-Verlag, 1974), p 195.